# Efficient Spike-Coding with Multiplicative Adaptation in a Spike Response Model

**Sander M. Bohte**
CWI, Life Sciences
Amsterdam, The Netherlands
`S.M.Bohte@cwi.nl`

## Abstract

Neural adaptation underlies the ability of neurons to maximize encoded information over a wide dynamic range of input stimuli. Recent spiking neuron models like the adaptive Spike Response Model implement adaptation as additive fixed-size fast spike-triggered threshold dynamics and slow spike-triggered currents. Such adaptation accurately models neural spiking behavior over a limited dynamic input range. To extend efficient coding over large changes in dynamic input range, we propose a multiplicative adaptive Spike Response Model where the spike-triggered adaptation dynamics are scaled multiplicatively by the adaptation state at the time of spiking. We show that, unlike the additive adaptation model, the firing rate in our multiplicative adaptation model saturates to a realistic maximum spike-rate regardless of input magnitude. Additionally, when simulating variance switching experiments, the model quantitatively fits experimental data over a wide dynamic range. Dynamic threshold models of adaptation furthermore suggest a straightforward interpretation of neural activity in terms of dynamic differential signal encoding with shifted and weighted exponential kernels. We show that when thus encoding rectified filtered stimulus signals, the multiplicative adaptive Spike Response Model achieves a high coding efficiency and maintains this efficiency over changes in the dynamic signal range of several orders of magnitude, without changing model parameters.

## 1 Introduction

The ability of neurons to adapt their responses to greatly varying sensory signal statistics is central to efficient neural coding [1, 2, 3, 4, 5, 6, 7]. Consequently, accurate models for the underlying mechanisms can provide insight into the nature of neural coding itself. For this, models of neural computation have to account for adaptation in a manner consistent with both experimental findings and notions of efficient neural coding.

Neural computation is often reduced to a linear-nonlinear-poisson (LNP) model: input signals are filtered, followed by a thresholding function that determines the firing probability of the neuron. In the Generalized Linear Model (GLM) [8] a refractory response in the form of a post-spike filter is added (figure 1). With experimental responses fitted to such LNP models, adaptation is found to adjust both the effective gain in the thresholding function and the linear filtering function [9, 10].

Neural adaptation responds primarily to changes in local stimulus contrast or, equivalently, to the local detection threshold [11, 12], and a number of theoretical studies account for adaptation from the perspective of optimal contrast estimation [12, 13]. Recent work by Ozuysal & Baccus [14] suggests that in a Linear-Nonlinear first-order Kinetics model (LNK), the gain depends on the local contrast of the filtered and rectified input signal.

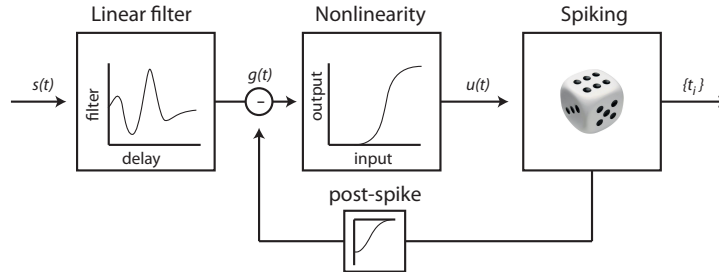

Figure 1: Generalized Linear Model (GLM) of neural computation.

With substantial spike-rate adaptation occurring on a time scale of just tens of milliseconds [4, 5], adapting neurons necessarily generate at most tens of spikes in that period. From an adaptive coding perspective, this implies that for a neuron's adaptation to be computable by downstream neurons, the adaptation effects have to be derivable from just the emitted spike-train. Spike-based models are thus central when accounting for adaptation as adaptive neural coding.

In variations of adaptive integrate-and-fire neurons [15, 16, 17], adaptation can be incorporated as a combination of two mechanisms: spike-triggered adaptation currents and a dynamical action-potential threshold. In such models, the adaptation mechanisms together increase the distance between the reversal potential and the threshold, effectively changing the gain of the neuron. The adaptive Spike Response Model [16, 17] in particular has been shown to be effective for modeling neural behavior in response to input currents with limited dynamic range [17]. On longer timescales, spike-triggered adaptation currents fit a power-law decay rather than an exponential decay, linking to observations of long-range power-law rate-adaptation [18, 19, 20, 21, 17].

Still, in spite of its success, the additive model of adaptation in adaptive Spike Response Model effectively changes neural gain with at most a fixed step-size, and thus cannot respond quickly to changes in signal variance that are large compared to this step-size. In particular, Brette [22] argues that adaptation modulation has to be multiplicative for neurons to respond with the same level of neural activity to drastic changes in dynamic range, as is observed experimentally (e.g. [4]).

We augment the adaptive Spike Response Model with multiplicative adaptation dynamics. We show that such a multiplicative adaptive Spike Response Model quantitatively matches neural responses in variance switching experiments and maximizes information transfer. Furthermore, we demonstrate that the model's effective gain responds to changes in either mean or variance of the filtered signal, similar to the LNK kinetic model in [14].

In the adaptive Spike Response Model, gain modulation derives from the difference between the adapted reversal potential and the dynamic threshold. This suggests a straightforward interpretation of spike-trains in terms of threshold-based detection of discernible signal levels in the rectified filtered input signal: *adaptive spike-coding*. We show how non-linear signal encoding with a multiplicative adaptive Spike Response Model maintains a high coding efficiency for stimuli that vary in magnitude over several orders of magnitude, unlike the additive version of the adaptive Spike Response Model. The coding efficiency is further comparable to the additive adaptive Spike Response Model when the adaptation step-size in the latter is optimized for the local dynamic range.

## 2 Spike-rate Adaptation in the Spike Response Model

We follow Naud *et al* [17] in modeling adaptation in an augmented Spike-Response Model [23]. In the adaptive Spike Response Model (aSRM), the dynamics of the (normalized) membrane-potential $V(t)$ are described as a sum of integrated input current $I(t)$ and spike-triggered currents $\eta(t)$:

$$V(t) = \int \phi(t-s)I(s)ds - \int \phi(t-s)\sum_{\{t_i\}} \eta(s-t_i)ds, \qquad (1)$$

where $\{t_i\}$ denotes the set of past emitted spikes, and the kernel $\phi(t)$ is a fast exponential low-pass filter on membrane currents:

$$\phi(t) = \phi_0 \exp\left(\frac{-t}{\tau_m}\right),$$

with $\tau_m$ determined by the membrane capacitance and conductance, and is typically on the order of several milliseconds [23, 17] .

The dynamical threshold is computed as the sum of a resting threshold $V_0$ and spike-triggered threshold dynamics $\gamma(t)$:

$$V_T(t) = V_0 + \sum_{\{t_i\}} \gamma(t - t_i). \tag{2}$$

Spikes are generated either deterministically when $V(t) - V_T(t)$ becomes positive, or stochastically following an inhomogeneous point process with conditional firing rate:

$$\lambda(t|V(t), V_T(t)) = \lambda_0 \exp\left(\frac{V(t) - V_T(t)}{\Delta V}\right), \tag{3}$$

where $\Delta V$ determines the slope of the exponential function; small values of $\Delta V$ approximate a neuron with a deterministic threshold. Naud *et al* [17] report that the threshold kernel $\gamma(t)$ is best fitted with an exponentially decaying function, whereas the shape of the spike-triggered current $\eta(t)$ depends on the type of neuron, and furthermore for longer timescales best fits a decaying power-law: $\eta(t - t_i) \propto (t - t_i)^{-\beta}$ for $t >> t_i$, with $\beta \approx 1$.

We can denote the effective neural threshold $\vartheta$ as the amount of input that will trigger a spike. In the adaptive Spike Response Model this amounts to the sum of the dynamic threshold, $V_T(t)$, and the (filtered) spike-triggered current: $\vartheta \propto V_T(t) + \int \phi(t - s) \sum_{\{t_i\}} \eta(s - t_i) ds$. We can move the reset response from (1) to the dynamic threshold (2) to obtain adaptation as the effective threshold dynamics $\vartheta(t)$:

$$\vartheta(t) = \vartheta_0 + \sum_{\{t_i\}} \left[ \gamma(t - t_i) + \int \phi(t - s)\eta(s - t_i)ds \right], \tag{4}$$

where $\vartheta_0 = V_0$ denotes the effective threshold for an inactive neuron. As the adaptation dynamics in this model are strictly additive, we will refer to it further as the *additive aSRM*.

The maximum effective threshold in the additive aSRM is limited by the maximum number of spikes that can be generated within the short time-window reported for variance adaptation. Effectively, the refractory period determines the upper bound for the adaptation step-size, and adaptation speed is upper-bounded by this value times the number of generated spikes.

## 2.1 Multiplicative Dynamic Adaptation

We propose a modification of the additive aSRM where the effective spike-triggered adaptation is not a fixed quantity but depends on the effective adaptation at the time of spiking. We include the multiplicative interaction in the aSRM by scaling the effective adaptation in (4) with the current adaptation value at the time of spiking:

$$\vartheta(t) = \vartheta_0 + \sum_{\{t_i\}} \vartheta(t_i) \left[ \gamma(t - t_i) + \int \phi(t - s)\eta(s - t_i)ds \right]. \tag{5}$$

For sparse spiking and adaptation response kernels that decay fairly rapidly to zero, such multiplicative adaptive threshold dynamics are approximately similar to the effective threshold dynamics in (4). For rapid signal variance transitions however, the multiplicative dynamics ensure that the effective threshold adaptation can rapidly range over multiple orders of magnitude.

The key difference in adaptation dynamics for the two aSRM models is illustrated in Figure 2. For a given spike-train, the respective adaptation magnitudes are plotted in Figure 2a , and the neural responses to different levels of step-size current injections are shown in Figure 2b. The additive aSRM responds to an increasing input current with a firing rate that is essentially only bounded by the refractory response; the firing rate in the aSRM with multiplicative adaptation saturates at a much lower value as the effective threshold catches up with the magnitude of the injected current.

## 2.2 Adaptive Spike-Coding

The interpretation of spike-triggered adaptation as dynamic neural gain in the Spike Response Model suggests a straightforward application to a spike-based neural coding model. Spike-rate adaptation

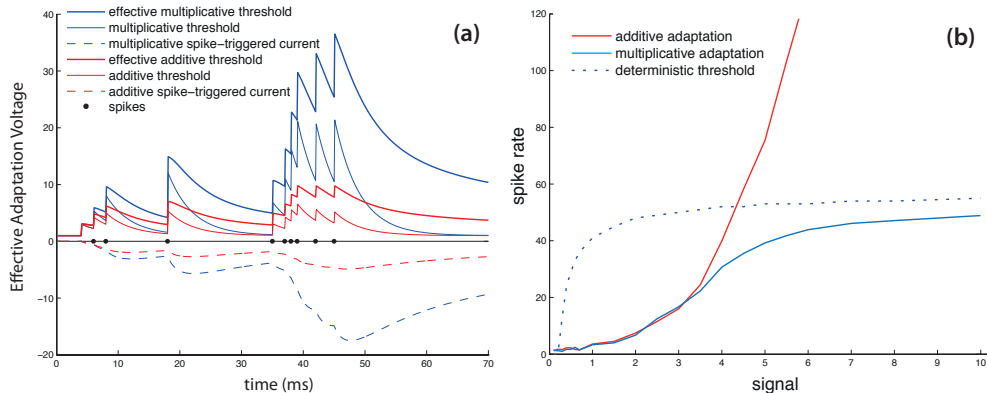

Figure 2: Illustration of multiplicative and additive threshold adaptation dynamics. (a) Effective adaptation as a sum of threshold dynamics (solid lines) and spike-triggered currents (dashed lines) given an input spike-train (black dots). Red lines correspond to additive adaptation dynamics, blue lines to multiplicative. (b) Firing rate as a function of signal strength. Red solid line is response for (stochastic) additive aSRM, blue solid line for the stochastic multiplicative aSRM; dotted blue line corresponds to a deterministic version of the multiplicative aSRM.

has been extensively studied from the point of view of optimal contrast estimation or signal threshold detection [13, 12]. In particular the notion of signal threshold detection suggests a simple model where individual spikes signal that the neuron has detected that its internally computed value has reached a level distinguishable from the local noise level [11].

Taking the standard Linear-Non-Linear model of neural computation, we follow Ozuysal & Baccus [14] in assuming that it is the rectified filtered version of the stimulus signal, $u(t)$, that is encoded by the spikes emitted by a neuron. We then define the Linear-Non-Linear-Adaptive-Thresholding (LNL-AT) model as greedy differential signaling: if the signal $u(t)$ exceeds a threshold value $\vartheta(t_i)$ at time $t_i$, a spike is generated communicating a scaled response kernel $\vartheta(t_i)\kappa(t-t_i)$ to downstream neurons. This response kernel is then also subtracted from the signal $u(t)$, and the dynamic threshold is updated to account for threshold adaptation (figure 3). In such greedy differential spike-coding, the signal $u(t)$ is effectively approximated as a sum of shifted and weighted response kernels:

$$\hat{u}(t) = \sum_{t_i < t} \vartheta(t_i)\kappa(t - t_i).$$

This adaptive spike-coding model corresponds to the multiplicative adaptive SRM in (5), where the filtered reset function $\int \phi(t-s)\eta(t)ds$ is interpreted as a response kernel $\kappa(t-t_i)$:

$$V(t) = \int \phi(t-s)I(s)ds - \sum_{t_i < t} \vartheta(t_i)\kappa(t - t_i), \tag{6}$$

$$= u(t) - \hat{u}(t),$$

$$\vartheta(t) = \vartheta_0 + \sum_{\{t_i\}} \vartheta(t_i)\gamma(t - t_i),$$

where spikes are generated when the membrane potential $V(t)$ exceeds the dynamic threshold $\vartheta(t)$. We let the threshold kernel $\gamma(t)$ fit a decaying power-law $\gamma(t - t_i) \propto (t - t_i)^{-\beta}$, and, to take advantage of temporal correlations, we model $\kappa(t)$ as an exponentially decaying kernel with time-constant $\tau_\kappa$ similar to the (average) correlation time of $u(t)$, $\kappa(t) = \exp(-t/\tau_\kappa)$ [24] (note that equation (5) implies that interchanging the behavior of $\eta(t)$ and $\gamma(t)$ does not change the SRM responses). Difference based neural coding models for spike-based neural coding have been noted in the context of probabilistic coding [25], and fast visual coding [26].

In this adaptive spike-coding model, each spike $t_i$ communicates a signal amount of magnitude $\vartheta(t_i)$. In particular for signal ranges where the firing rate saturates, the effective signal magnitude per spike grows linearly with signal size. This is depicted in figure 4, for a neuron with a stochastic

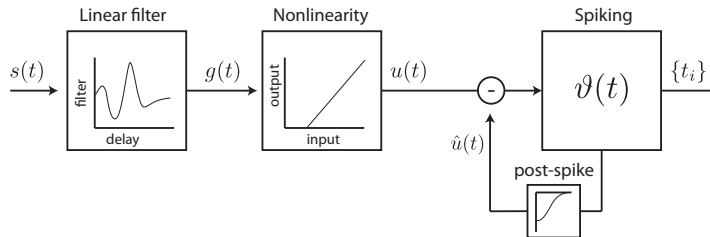

Figure 3: The Linear-Non-Linear-Adaptive-Thresholding (LNL-AT) model.

threshold (large $\Delta V$ in (3); figure 4a) and for a neuron with a deterministic threshold (small $\Delta V$ in (3); figure 4b). Plotted is the neural behavior in response to a range of step-size increases in the signal $u(t)$, where firing rate and effective adapted threshold are measured two seconds after the step-size signal increase. The average firing rate shows the traditional saturation of neural response with increasing signal size. However, the effective adapted threshold increases linearly with signal size, paralleling the $u = u$ signal identity.

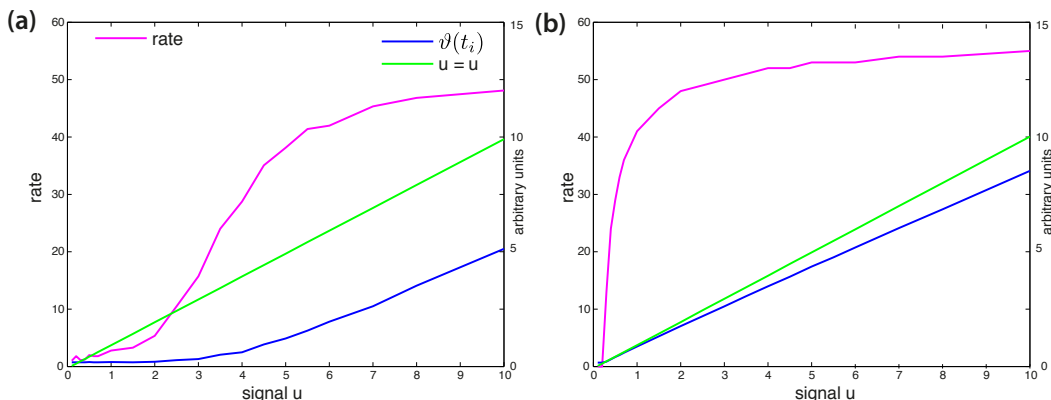

Figure 4: Effective adapted threshold $\vartheta(t_i)$ (right axis) and firing rate (left axis) as a function of signal size $u$ (a) stochastic multiplicative aSRM; (b) deterministic multiplicative aSRM.

# 3 Results

We demonstrate how the multiplicative aSRM quantitatively fits with key findings on adaptation in experimental data.

## 3.1 Variance Switching

The neural responses to variance switching [4, 5] in sensory signals are considered central evidence for the information maximizing effect of adaptation, and also demonstrate the fast timescale of (initial) adaptation. In these key experiments, recordings are obtained from the blowfly's H1 neuron, and its responses are measured to a repeated change in perceived velocity variance. Signal variance is repeatedly scaled from $\sigma_1$ to $\sigma_2 = 10 * \sigma_1$, with a cycle time $T$. As the cycle-time $T$ is increased, the effective time constant of adaptation grows (as measured by fitting an exponent on the initial segment of the decaying curve). This time-constant of adaptation shows scale-free behavior: when normalizing for the interval time $T$, the neural response curves overlap, and there is linear relationship between cycle-time $T$ and effective adaptation time constant $\tau$. As reported in [27], the additive aSRM is only able to match these findings qualitatively for a limited change in variance.

As in [4, 5], we generated random white noise within an interval enclosed by $[-\sigma_i, \sigma_i]$, for different values of the variance $\sigma_i$ (1 and 10 respectively). This signal was filtered with filters obtained by the GLM-model [8] on the original data from [4]. We fed the thus filtered and rectified signal into the multiplicative aSRM and optimized the model parameters using exhaustive line-search.

The optimized multiplicative aSRM exhibits both the same firing behavior and the same relationship between normalized switching interval and normalized firing rate as the experimental data in [5] (Figure 5b,c). Furthermore, characterizing the input-output relationship as in [5] recovers the same overlapping response-curves after normalizing the projected velocity signal for the scaled variance. The fitted adaptation decay time-constant $\tau$ also closely matches the experimental data [5] (Figure 5e, simulation: red circles, data: black circles). Changing the dynamic range for both $\sigma_1$ and $\sigma_2 = 10 * \sigma_1$ by a factor of 10 did not change the relationship (green dots). We also characterized the signal versus firing rate response for three scaled versions of the same velocity signal, with scaling factors 1, 2 and 3, similar to [4] (open markers, Figure 5f). As in [4], the adapted signal-rate response curves also overlap after normalizing the signal for the scaled variance (solid markers, Figure 5f). Multiplicative effective adaptation thus maximizes the transmitted information as in [4, 5].

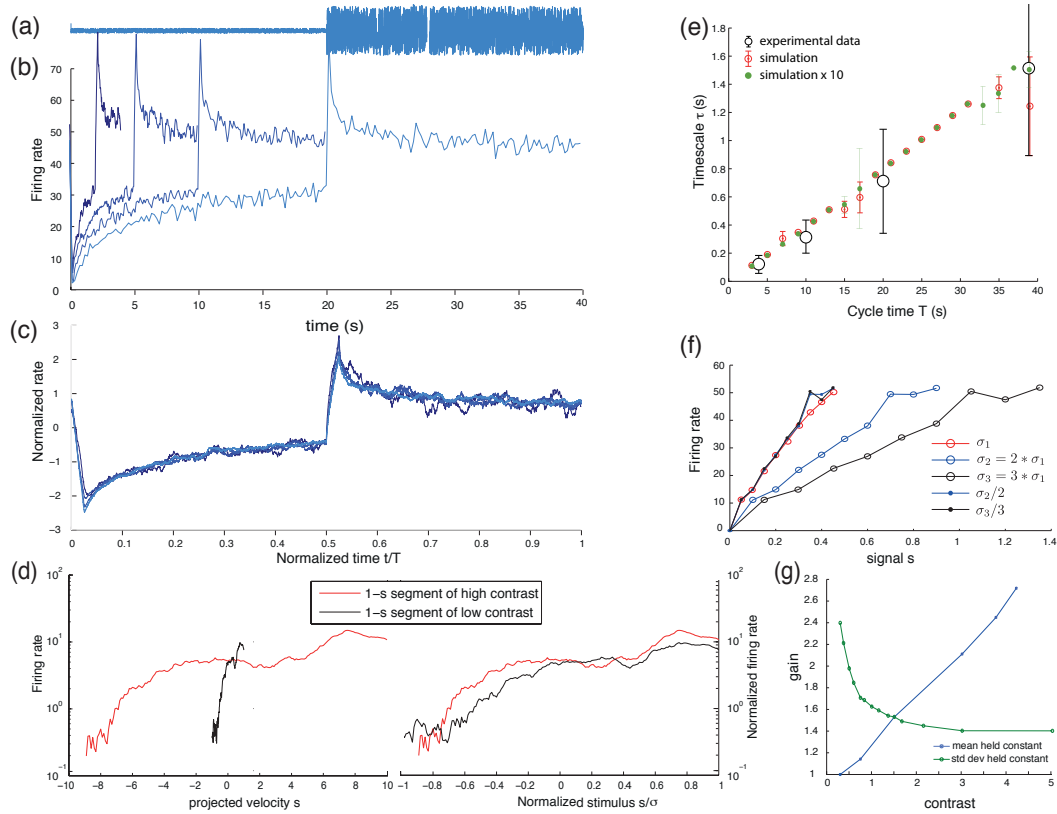

Figure 5: Variance switching. (a) variance of sensory input is switched with a fixed cycle time. (b) The aSRM neuron adapts its firing rate after each switch. Switching responses for different cycle times are overlapped. (c) The response curves for various cycle times overlap when time is normalized for cycle time $T$. (d) Input-output relationship derived from 1-s-wide time windows in the two signal variance conditions: left projected velocity signal $s$ vs normalized firing rate, right, projected velocity signal $s$ normalized by respective variance $\sigma$. (e) Relationship between fitted adaptation timescale $\tau$ as a function of cycle time $T$. Red circles simulation data; black circles experimental data from [5]. Green dots are simulation data for switching signals multiplied by a factor 10. (f) Simulation response to signal scaled by factors $\sigma_1 = 1$, $\sigma_2 = 2$, $\sigma_3 = 3$ (open markers), and responses rescaled by signal scale factor (solid markers). (g) Effective gain $(1/\vartheta(t))$ in the multiplicative aSRM neuron as a function of contrast, for signal $u$ with mean held constant and variance varied (blue line), and variance held constant and mean varied (green line). For the experiments, resting threshold $\vartheta_0$ was set to $0.008$, spike-triggered adaptation currents decayed with a power-law constant of $\beta = 1.15$, as $3.5(t - t_i + 0.7)^{-\beta}$ and response kernels as $2.5 \exp(-t/9)$ (time $t$ in ms).

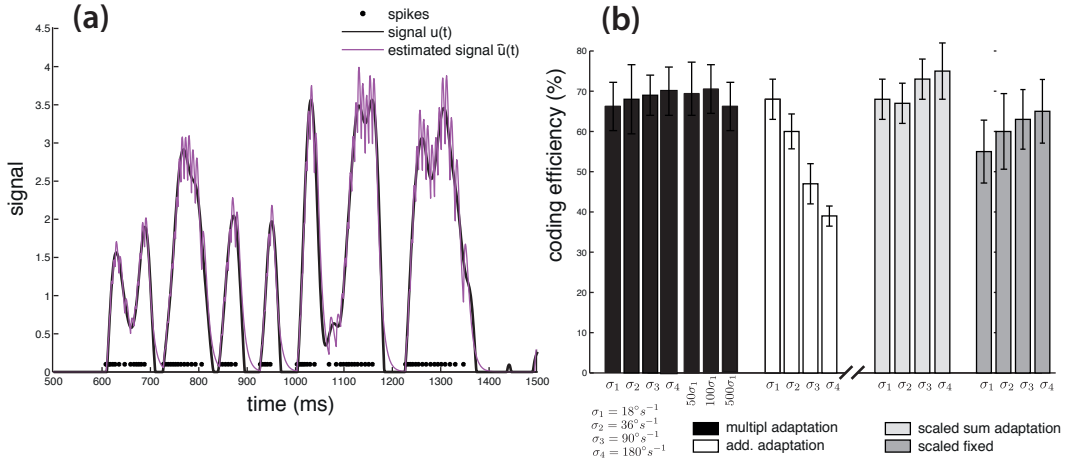

Figure 6: Multiplicative Spike-Coding: (a) illustration of stimulus encoding as a sum of shifted and weighted response kernels. Black dots denote spike-times, black solid line the signal $u(t)$, and magenta the approximated signal $\hat{u}(t)$. (b) Computed coding efficiency. Information rate $R_{\text{info}}$ was computed, with effective signal and noise bandwidth cutoff at 50Hz (matching the original stimulus signal). Coding efficiency was computed by dividing $R_{\text{info}}$ by the spike-train entropy rate $S/T$ [28] for a timing precision of 1 ms. Model parameters for the multiplicative aSRM are as in Figure 4. Note that for the grey and light-grey bars refer to the left, parameters are optimized for each $\sigma$ value individually.

For adaptation to relate to contrast, loosely defined as the ratio of (local) standard deviation $\sigma$ and local average signal $\bar{u}$, $\sigma/\bar{u}$ (and thus detection threshold), it should respond accordingly to changes in not just variance but also in changes to mean (rectified) signal magnitude. Ozuysal & Baccus [14] show that this property holds for their kinetic model of gain modulation, which also closely matches experimental data. In the kinetic model, effective gain scales linearly with standard deviation when all other signal statistics are held constant, and similarly with $1/\bar{u}$; in simulations, where effective gain in computed as $1/\vartheta(t)$, we find that the multiplicative aSRM shares this property (Figure 5g).

### 3.2  H1 encoding/decoding

With multiplicative effective adaptation responding to contrast changes, we can examine the effectiveness of the corresponding neural coding model. For this, we use the original blowfly data from Brenner *et al* [4], consisting of velocity stimulus profiles presented to the blowfly, where the velocity stimulus is scaled with factors of $\sigma_1 = 18^\circ s^{-1}, \sigma_2 = 2\sigma_1 = 36^\circ s^{-1}, \sigma_3 = 90^\circ s^{-1}$ and $\sigma_4 = 180^\circ s^{-1}$. We examine how well multiplicative adaptive neural coding approximates the rectified filtered signal, as compared to such neural coding with the additive aSRM.

We filter each version of this velocity stimulus with the filter obtained using GLM optimization on the velocity stimulus with variance $\sigma_1$ and optimize the parameters in both aSRM models for condition $\sigma_1$, using deterministic thresholds. Adaptation was highly robust for the parameters, provided we chose an exponential response kernel with time-constant 10ms to match the correlation time of the filtered signal. We further tuned the resting threshold $\vartheta_0$ and magnitude of the power-law adaptation kernel $\gamma$ so that the average firing rate matched the experimental data at least for the $\sigma_1$ signal. An example of stimulus encoding with multiplicative adaptive neural coding is shown in figure 6a.

We compare coding efficiency for the multiplicative aSRM and for the additive aSRM for a spike precision of 1ms [28], applying the model optimized for condition $\sigma_1$ to all four stimulus conditions $\sigma_1, \sigma_2, \sigma_3, \sigma_4$, and, for the multiplicative aSRM additionally for the conditions $50 \times \sigma_1, 100 \times \sigma_1, 500 \times \sigma_1$. Relative coding efficiencies are plotted in figure 6b, black and white bars. We see that the multiplicative aSRM maintains a high coding efficiency over the entire dynamic range, even for the $500 \times \sigma_1$ stimulus condition. The dynamic range of the additive aSRM however is insufficient to encode the wide dynamic range of the original data. Similar to the experiment in [4], we find

that the firing rate for the multiplicative aSRM signal encoding remains approximately stable for all stimulus conditions, with a firing rate of $55 \pm 5$ spikes/s, without changing any parameters. The firing rate for the additive aSRM increases from a (matched) firing rate of 55 spikes/s for the $\sigma_1$ stimulus, to over 180 spikes/s for the $\sigma_4$ stimulus.

We also compare against the additive aSRM and neural coding with a non-adaptive, fixed response kernel SRM, with the magnitude of the response-kernel (equivalent to $\vartheta_0$) optimized for the local variance such that for each stimulus, the firing rate for these models matches that of the multiplicative aSRM. This is shown in the light grey (scaled additive aSRM) and dark grey (scaled non-adaptive SRM) bars in figure 6b. The coding efficiency for multiplicative aSRM is close to that of locally rescaled additive aSRM's, and exceeds locally rescaled non-adaptive coding.

## 4 Discussion

We showed how a multiplicative model of neural adaptation in the Spike Response Model can account quantitatively for key experimental adaptation data. When interpreting the fast adaptation component as the manifestation of a greedy signal encoding scheme, we further showed that multiplicative adaptation allows the Spike Response Model to achieve high coding efficiency for signals with dynamic ranges that change over several orders of magnitude, without changing parameters. Just as the H1 blowfly neuron, the multiplicative aSRM uses a near-constant firing rate for the widely varying dynamic range in the different stimulus conditions.

The ubiquity of adaptation in neural systems and notions of synaptic facilitation and depression suggest that gain modulation could possibly be decoded in a receiving neuron by adaptively scaling the size of the post-synaptic response. Although Series [29] argues that a number of visual percepts are consistent with decoding neurons being "unaware" of presynaptic adaptation, the presence or absence of such coupled adaptation can be considered as a form of spectral filtering [30]. As we have shown, a key advantage of accounting for gain modulation in spike-based neural coding is that it greatly extends the neuron's dynamic range, and may allow for instance implicit spike-based probabilistic computation as in [31] to scale to multiple layers.

From a biological perspective, it may seem implausible to let threshold dynamics and spike-triggered adaptation currents scale with vast changes in dynamic range. However, as noted in [17], there is a theoretical link between spike-triggered plasticity like spike-timing dependent plasticity and spike-triggered adaptation [32]. That is, scaling of synaptic weights could complement adaptation to large changes in dynamic range. The multiplicative adaptive Spike Response Model also captures only part of the first-order dynamics in the LNK model in [14], and does not account for variance-dependent changes in temporal filtering (e.g. [9]). Thus, spike-based adaptation of the response kernel could likely further improve the coding efficiency.

The multiplicative adaptive Spike Response Model provides a spike-based account for gain modulation, which can easily be reconstructed by post-synaptic neurons as a function of the received spike-train. It thus provides an effective neuron model for dynamical spiking neural networks, resolving for instance stability problems in spiking reservoir computing approaches.

**Acknowledgement.** The author thanks Hao Wang for assistance with the simulations, and Jaldert Rombouts, Kausik Lakshminarasimhan and Hao Wang for helpful suggestions.

## References

[1] S B Laughlin. The role of sensory adaptation in the retina. *The Journal of experimental biology*, 146:39–62, September 1989.

[2] S.M. Smirnakis, M.J. Berry, D.K. Warland, W. Bialek, and M. Meister. Adaptation of retinal processing to image contrast and spatial scale. *Nature*, 386(6620):69–73, 1997.

[3] M.J. Wainwright. Visual adaptation as optimal information transmission. *Vision Research*, 39(23):3960–3974, 1999.

[4] N. Brenner, W. Bialek, and R. de Ruyter van Steveninck. Adaptive rescaling maximizes information transmission. *Neuron*, 26(3):695–702, 2000.

[5] A.L. Fairhall, G.D. Lewen, W. Bialek, and R.R. de Ruyter van Steveninck. Efficiency and ambiguity in an adaptive neural code. *Nature*, 412(6849):787–792, 2001.

[6] O Schwartz and E P Simoncelli. Natural signal statistics and sensory gain control. *Nature Neuroscience*, 4(8):819–25, 2001.

[7] T. Hosoya, S.A. Baccus, and M. Meister. Dynamic predictive coding by the retina. *Nature*, 436(7047):71–77, 2005.

[8] J.W. Pillow, L. Paninski, V.J. Uzzell, E.P. Simoncelli, and E.J. Chichilnisky. Prediction and decoding of retinal ganglion cell responses with a probabilistic spiking model. *Journal of Neuroscience*, 25(47):11003–13, 2005.

[9] S. Baccus and M. Meister. Fast and slow contrast adaptation in retinal circuitry. *Neuron*, 36(5):909–19, 2002.

[10] S. Hong, B.N. Lundstrom, and A.L. Fairhall. Intrinsic gain modulation and adaptive neural coding. *PLoS Computational Biology*, 4(7), 2008.

[11] H P Snippe and J H van Hateren. Recovery from contrast adaptation matches ideal-observer predictions. *Journal of the Optical Society of America. A, Optics, image science, and vision*, 20(7):1321–1330, 2003.

[12] H P Snippe, L Poot, and J H Van Hateren. Asymmetric dynamics of adaptation after onset and offset of flicker. *Journal of Vision*, pages 1–12, 2004.

[13] M. DeWeese and A. Zador. Asymmetric dynamics in optimal variance adaptation. *Neural Comp*, 10(5):1179–1202, 1998.

[14] Y. Ozuysal and S.A. Baccus. Linking the Computational Structure of Variance Adaptation to Biophysical Mechanisms. *Neuron*, 73(5):1002–1015, March 2012.

[15] R Harris, D C O'Carroll, and S B Laughlin. Contrast gain reduction in fly motion adaptation. *Neuron*, 28(2):595–606, 2000.

[16] R. Jolivet, A. Rauch, H.R. Lüscher, and W. Gerstner. Predicting spike timing of neocortical pyramidal neurons by simple threshold models. *Journal of computational neuroscience*, 21(1):35–49, 2006.

[17] R Naud. *The Dynamics of Adapting Neurons*. PhD thesis, EPFL Lausanne, 2011.

[18] P.J. Drew and LF Abbott. Models and properties of power-law adaptation in neural systems. *Journal of neurophysiology*, 96(2):826, 2006.

[19] Z. Xu, JR Payne, and ME Nelson. Logarithmic time course of sensory adaptation in electrosensory afferent nerve fibers in a weakly electric fish. *Journal of neurophysiology*, 76(3):2020, 1996.

[20] B.N. Lundstrom, M.H. Higgs, W.J. Spain, and A.L. Fairhall. Fractional differentiation by neocortical pyramidal neurons. *Nature neuroscience*, 11(11):1335–1342, 2008.

[21] B. Wark, A. Fairhall, and F. Rieke. Timescales of inference in visual adaptation. *Neuron*, 61(5):750–761, 2009.

[22] R. Brette. Spiking models for level-invariant encoding. *Front. in Comp. Neurosc.*, 5, 2011.

[23] W. Gerstner and W. Kistler. *Spiking Neuron Models: Single Neurons, Populations, Plasticity*. Cambridge University Press, 2002.

[24] M. Buiatti and C. van Vreeswijk. Variance normalisation: a key mechanism for temporal adaptation in natural vision? *Vision Research*, 43(17):1895–1906, August 2003.

[25] S. Deneve. Bayesian spiking neurons I: inference. *Neural computation*, 20(1):91–117, 2008.

[26] P. Lichtsteiner, C. Posch, and T. Delbruck. A 128× 128 120 db 15 $\mu$s latency asynchronous temporal contrast vision sensor. *Solid-State Circuits, IEEE Journal of*, 43(2):566–576, 2008.

[27] C Pozzorini, R Naud, S Mensi, and W Gerstner. Multiple timescales of adaptation in single neuron models. In *Front. Comput. Neurosci. Conference Abstract: BCCN*, 2010.

[28] F. Rieke, D. Warland, and W. Bialek. *Spikes: exploring the neural code*. The MIT Press, 1999.

[29] P. Seriès, A.A. Stocker, and E.P. Simoncelli. Is the Homunculus "Aware" of Sensory Adaptation? *Neural Computation*, 21:3271–3304, 2009.

[30] S.M. Bohte and J.O. Rombouts. Fractionally Predictive Spiking Neurons. In *Advances in Neural Information Processing Systems (NIPS) 23*, pages 253–261. The MIT Press, 2010.

[31] W.J. Ma, J.M. Beck, P.E. Latham, and A. Pouget. Bayesian inference with probabilistic population codes. *Nature neuroscience*, 9(11):1432–1438, November 2006.

[32] G. Hennequin, W. Gerstner, and J.P. Pfister. Stdp in adaptive neurons gives close-to-optimal information transmission. *Front. in Comp. Neurosc.*, 4, 2010.

